# TPR: Topology-Preserving Reservoirs for Generalized Zero-Shot Learning

**Hui Chen[1]\*, Yanbin Liu[2] , Yongqiang Ma[1] , Nanning Zheng[1]†, Xin Yu[3]**
[1]National Key Laboratory of Human-Machine Hybrid Augmented Intelligence,
National Engineering Research Center of Visual Information and Applications,
and Institute of Artificial Intelligence and Robotics, Xi'an Jiaotong University,
[2]Auckland University of Technology,
[3]The University of Queensland

## Abstract

Pre-trained vision-language models (VLMs) such as CLIP have shown excellent performance for zero-shot classification. Based on CLIP, recent methods design various learnable prompts to evaluate the zero-shot generalization capability on a ***base-to-novel*** setting. This setting assumes test samples are already divided into either base or novel classes, limiting its application to realistic scenarios. In this paper, we focus on a more challenging and practical setting: ***generalized zero-shot learning*** (GZSL), *i.e.*, testing with no information about the base/novel division. To address this challenging zero-shot problem, we introduce two unique designs that enable us to classify an image without the need of knowing whether it comes from seen or unseen classes. *Firstly*, most existing methods only adopt a single latent space to align visual and linguistic features, which has a limited ability to represent complex visual-linguistic patterns, especially for fine-grained tasks. Instead, we propose a dual-space feature alignment module that effectively augments the latent space with a novel attribute space induced by a well-devised attribute reservoir. In particular, the attribute reservoir consists of a static vocabulary and learnable tokens complementing each other for flexible control over feature granularity. *Secondly*, finetuning CLIP models (*e.g.*, prompt learning) on seen base classes usually sacrifices the model's original generalization capability on unseen novel classes. To mitigate this issue, we present a new topology-preserving objective that can enforce feature topology structures of the combined base and novel classes to resemble the topology of CLIP. In this manner, our model will inherit the generalization ability of CLIP through maintaining the pairwise class angles in the attribute space. Extensive experiments on twelve object recognition datasets demonstrate that our model, termed Topology-Preserving Reservoir (TPR), outperforms strong baselines including both prompt learning and conventional generative-based zero-shot methods.

## 1 Introduction

Young children often exhibit a remarkable capacity to identify novel visual objects only based on verbal descriptions provided by their caregivers. This phenomenon has spurred considerable interest in developing learning models with similar feats, known as zero-shot learning (ZSL). Early ZSL works [1, 2] trained a model on the seen base classes and evaluated the generalization performance on novel unseen classes. This can be done by aligning the visual features and textual descriptors

---

*\*Work done while visiting The University of Queensland.*
*†Corresponding author: nnzheng@mail.xjtu.edu.cn.*

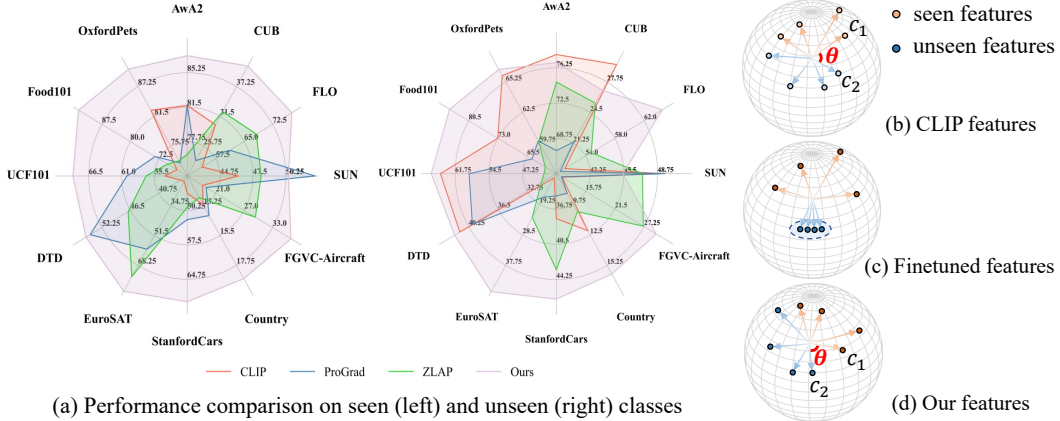

Figure 1: (a) In the challenging and realistic *generalized zero-shot learning* (GZSL) setting, our method significantly outperforms the state-of-the-art methods on both seen and unseen classes. (b-d) Finetuning CLIP will lead to the weak generalization problem [8] on unseen classes. We tackle this problem by inheriting the topology of CLIP feature space by maintaining the pairwise angles.

into a shared latent space, where the intrinsic visual-linguistic relationship is extracted for zero-shot classification at test time. However, this conventional ZSL setting assumes test examples only come from novel classes, limiting its application to realistic scenarios where both base and novel classes need to be classified. Therefore, a more realistic setting *generalized zero-shot learning* (GZSL) [3, 4] has been proposed to recognize both base and novel classes without knowing the base/novel division. Recent GZSL methods such as CE [5], LSA [6] and ZLAP [7] design various generative models, which generate the missing visual features for unseen novel classes and then conduct joint base and novel classification.

The emergence of large vision-language models (VLMs) such as CLIP [9] have demonstrated good potential for ZSL. For example, CLIP is trained on 400 million image-text pairs, which can effectively capture the visual-linguistic links essential for ZSL. After training, the model can be applied for zero-shot classification using a hand-crafted prompt, *e.g.*, 'a photo of <class>'. Starting from CLIP, recent works design diverse learnable prompts (*e.g.*, conditional prompt [8], Multi-modal prompt [10], and self-regulating prompt [11]) to improve CLIP's performance for downstream zero-shot tasks. We find that these prompt methods adopt a *base-to-novel* setting to evaluate their zero-shot generalization capability, similar to the conventional ZSL setting. In particular, they classify the base and novel classes separately, with the strong assumption that test samples have already been divided into either base or novel classes. Driven by the practicality of zero-shot learning, this paper focuses on a more realistic setting of *generalized zero-shot learning* (GZSL) under the VLM context.

To tackle the challenging GZSL problem, we introduce a Topology-Preserving Reservoir (TPR) model to effectively unleash the generalization potential of VLMs for the simultaneous classification of base and novel categories. To achieve the goal, our proposed TPR embraces two core novel designs: a dual-space feature alignment module and a feature semantic topology preserving objective. Considering that most previous methods establish the visual-linguistic relations by employing a single shared latent space [7, 11, 8, 6], the complex and fine-grained patterns cannot be effectively captured. To mitigate this key issue, we present a dual-space feature alignment module by enhancing the latent space with a representative attribute space, which is constructed from a well-devised attribute reservoir. The reservoir is designed to contain both static and learnable vocabulary tokens. In this fashion, both prior knowledge and task-specific information can be extracted, enriching the feature representations and avoiding overfitting to a single task.

Moreover, recent works [8] identify the weak generalizability problem of prompt learning on VLMs, *i.e.*, the learned prompts on seen classes do not often generalize well to unseen classes (*e.g.*, in Fig. 1 (a) (right), ProGrad [12] underperforms CLIP on unseen classes). To address this problem, we propose a new topology-preserving objective (Fig. 1 (b-d)) to maintain the semantic topology structure of the combined seen and unseen[3] classes by referring to the original CLIP embeddings.

Specifically, we adopt the Pearson correlation coefficient to constrain the variation of angles between pairwise categories before and after CLIP finetuning. As a result, after finetuning, our model still inherits the good generalization ability of CLIP, without suffering from the weak generalization problem. This can be shown by the superior performance on the unseen classes in Fig. 1(a)(right).

Extensive experiments conducted on twelve object recognition datasets, including both traditional GZSL benchmarks and prompt learning benchmarks, demonstrate that our proposed method significantly improves the recognition performance on unseen classes over our baselines, and outperforms the state-of-the-art on eleven datasets. Our contributions are summarized as follows:

- Different from the traditional *base-to-novel* setting, our work emphasizes a challenging yet practical *generalized zero-shot* learning problem for VLMs, without knowing the division of the base and novel categories. In this scenario, our proposed Topology-Preserving Reservoir model significantly improves the recognition performance compared to prior arts.

- We introduce a dual-space feature alignment module. It enhances the latent space, which is shared by visual and textual features, with a representative attribute space constructed from an attribute reservoir. In this manner, we essentially improve the representativeness of the latent space, leading to better fine-grained alignment between visual and textual features.

- We introduce a new feature semantic topology-preserving objective that is designed to maintain the semantic topology structure of finetuned features. In this fashion, finetuned features will not severely overfit to seen classes. Thus, we can effectively preserve the generalization capability of VLMs on unseen categories.

## 2    Related Works

**Zero-shot Learning.** Zero-shot Learning (ZSL) [2, 13, 14, 15] aims to recognize unseen objects by leveraging auxiliary knowledge such as tags, attributes, or textual descriptions to bridge the gap between seen and unseen classes. Generalized Zero-shot Learning (GZSL) [16, 17, 18] extends the scope of ZSL by considering a more realistic scenario where both seen and unseen classes are present during testing. Several methods [19, 20, 21, 22, 23, 16, 24] seek to learn a latent space where visual and linguistic data are aligned and the inference is performed by searching which class has the highest similarity score. Besides, generative models [25, 26, 27, 7, 6, 5, 28] like GANs have been explored for ZSL/GZSL due to their superior performance. These models generate synthetic samples for unseen classes based on their semantic descriptions. In the Large Language Model (LLM) era, many works [24, 29, 30] focus on generating improved visual-linguistic features for better alignment. While most methods default to utilizing attributes as semantic embeddings, attribute annotation poses scalability challenges for large-scale datasets. Moreover, attribute annotation exhibits subjectivity [31], leading to perceptible discrepancies among different annotators.

**Vision-Language Models.** Vision-Language Models (VLMs) aim to bridge the semantic gap between visual and textual modalities, enabling tasks such as image captioning, visual question answering, and image-text retrieval. Inspired by the success of self-supervised learning [32], vision-language pretraining has emerged as a powerful paradigm for learning rich representations of images and text. Recent VLMs, such as CLIP [9], ALIGN [33], BLIP [34], VLMO [35], CoCa [36] and FLIP [37], learn powerful joint representations by using contrastive learning on large amounts of paired vision-language data. After pretraining on 400M pairs of data, CLIP constructs image classifiers using the class names of the target dataset in a zero-shot manner and achieves superior performance. Despite the direct applicability of VLMs to zero-shot recognition, empirical observations show suboptimal performance on fine-grained tasks [9]. In this work, we propose to align multimodal representations in a dual-space to improve the fine-grained representation ability of VLMs for GZSL.

**Prompt Learning in VLMs.** Prompt learning has gained traction in natural language processing as a powerful approach for adapting pre-trained language models to new tasks with minimal supervision [38, 39]. In prompt learning, task-specific prompts or templates are designed to guide the language model to generate outputs tailored to a particular task, enabling effective adaptation to diverse downstream applications. Recent research [40, 8] has extended the concept of prompt learning to VLMs. By providing task-specific prompts that incorporate both visual and textual cues, VLMs can seamlessly integrate knowledge from multiple modalities and generalize to unseen tasks with limited labeled data [10, 41, 11, 12]. For example, CoOp [40] finetunes the pretrained CLIP model by

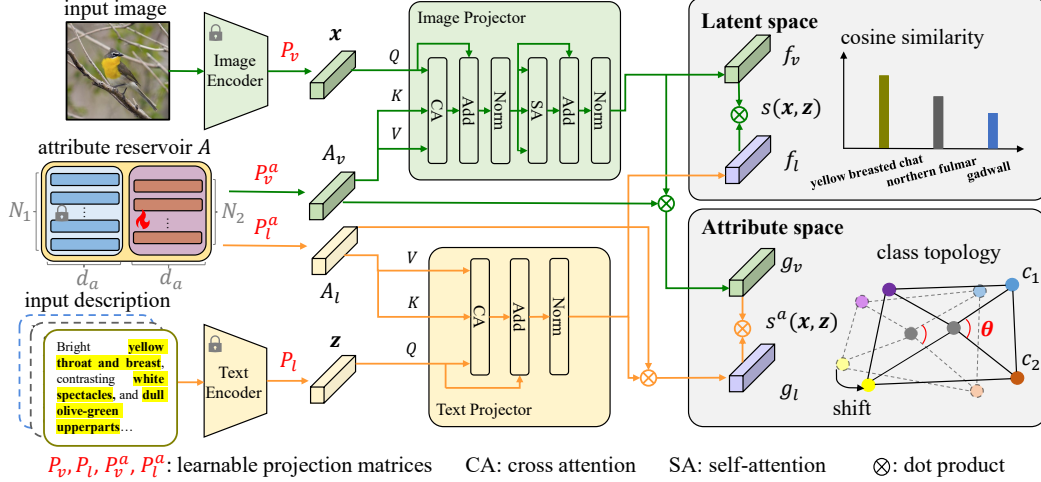

Figure 2: Overview of our TPR framework. The latent space directly aligns visual and linguistic features extracted from frozen VLMs. To augment latent space for fine-grained visual-textual pattern mining, we devise a novel attribute reservoir for constructing a new attribute space. The reservoir consists of both static and learnable vocabulary tokens, enabling flexible exploration and control of feature granularity for the GZSL task. Furthermore, we propose a topology-preserving objective to keep the generalization capability of VLMs, mitigating the weak generalization problem [8].

inserting learnable context vectors into a fixed textual template. However, recent work [8] identified a weak generalizability problem of prompt learning: the learned prompt is not generalizable to wider unseen classes. In this work, we devise a novel topology-preserving objective to tackle this problem.

## 3 Methodology

We propose a Topology-Preserving Reservoir (TPR) framework (Fig. 2) to unleash the generalization capability of VLMs for GZSL. Specifically, TPR has two unique designs on top of VLMs: (1) *Dual-space feature alignment module* to strengthen feature discriminability by aligning visual and linguistic features in both latent space and attribute space (constructed by the well-devised attribute reservoir); (2) *Feature semantic topology-preserving objective* to maintain the generalization capability of VLMs through preserving both seen and unseen class topology before and after fine-tuning.

**Problem Formulation.** Different from conventional ZSL or base-to-novel setting, GZSL needs to address the challenge of recognizing both seen and unseen classes without knowing the seen/unseen division at test time. During training, only samples from the seen classes are available. Formally, given a dataset $\mathcal{S} = \{(x_i^s, y_i^s, z_i^s) | x_i^s \in \mathcal{X}^s, y_i^s \in \mathcal{Y}^s, z_i^s \in \mathcal{Z}^s\}_{i=1}^{n_1}$ consist of $n_1$ samples, where $x_i^s, y_i^s, z_i^s$ denote the visual feature, class label, and textual description feature of the $i$-th seen image, respectively. Meanwhile, another dataset $\mathcal{U} = \{(x_j^u, y_j^u, z_j^u) | x_j^u \in \mathcal{X}^u, y_j^u \in \mathcal{Y}^u, z_j^u \in \mathcal{Z}^u\}_{j=1}^{n_2}$ contains $n_2$ samples, where $x_j^u, y_j^u, z_j^u$ represent the visual feature, class label, and textual description feature of the $j$-th unseen image, respectively. The seen and unseen label sets are disjoint: $\mathcal{Y}^s \cap \mathcal{Y}^u = \emptyset$. Following common practice in GZSL, the seen dataset $\mathcal{S}$ is split into a training set $\mathcal{D}_{tr}^s$ and a test set $\mathcal{D}_{te}^s$, while the unseen dataset $\mathcal{U}$ constitutes the test set $\mathcal{D}_{te}^u$. Then, the model is trained on $\mathcal{D}_{tr}^s$ and evaluated on the union set $\mathcal{D}_{te}^s \cup \mathcal{D}_{te}^u$. In the following, $s$ and $u$ will be omitted for simplicity.

### 3.1 Dual-Space Feature Alignment

**Attribute Reservoir Construction.** We devise the attribute reservoir to construct a new attribute space, which can exploit the meticulous features overlooked by a simple latent space. This further facilitates the effective mining of complex visual-linguistic patterns for better GZSL. The reservoir design takes into account both the generalization capability and task-specific adaptation. Initially, we curate an extensive array of attribute terms sourced from diverse literature repositories, including CUB [42], MIT-States [43], MAD [31], VAW [44] and LSA [45], which collectively delineate the shape, color, motion, material, texture, and part of an object. After eliminating redundancies, we

obtain a **base attribute vocabulary** of size $N_1$, including attributes such as *washing up*, *on stick*, and *pinstriped*. Subsequently, we employ a pre-trained LLM [46] to extract features of this base attribute vocabulary for GZSL, obtaining $A_1 \in \mathbb{R}^{N_1 \times d_a}$. The attribute vocabulary covers extensive attribute repositories to facilitate the generalization to unseen classes, but it is impossible to exhaustively include all conceivable attributes lying in image and textual data. Therefore, we introduce the flexible **learnable attribute tokens**, denoted as $A_2 \in \mathbb{R}^{N_2 \times d_a}$, to augment the base attribute vocabulary. These tokens have two functions: (1) they learn complementary attribute knowledge absent in the base attribute vocabulary in a data-driven manner, and (2) they incorporate the task-specific information into the reservoir for better downstream task adaptation. Eventually, the two components of reservoir are concatenated together to form our attribute reservoir $A \in \mathbb{R}^{N \times d_a}$, where $N = N_1 + N_2$.

**Multi-Modality Encoding.** Given an input image, we initially utilize the pre-trained CLIP image encoder to extract its visual feature $x \in \mathbb{R}^{1 \times d}$. Subsequently, we project the visual feature into both a latent space and an attribute space (*i.e.*, dual-space). To achieve this, we transform the visual feature and attribute reservoir into the same dimension with two linear layers ($P_v, P_v^a$):

$$x \leftarrow xP_v \in \mathbb{R}^{1 \times d}, A_v = AP_v^a \in \mathbb{R}^{N \times d}. \tag{1}$$

The visual feature $x$ is then encoded into the dual-space:

$$x' = \text{Attn}(x, A_v, A_v), x'' = \text{Attn}(x', x', x'), f_v = x'' + x \in \mathbb{R}^{1 \times d}, g_v = f_v A_v^T \in \mathbb{R}^{1 \times N}, \tag{2}$$

where $\text{Attn}(\cdot, \cdot, \cdot)$ is the attention function [47]: $\text{Attn}(\text{Q}, \text{K}, \text{V}) = \texttt{softmax}(QK^T/\sqrt{d})V$, $f_v$ denotes the visual feature encoded in the latent space, and $g_v$ represents the visual feature encoded in the attribute space. On the text side, given the textual description corresponding to the image, we utilize the CLIP text encoder to extract the linguistic feature $z \in \mathbb{R}^{1 \times d}$. Subsequently, we transform $z$ and the attribute reservoir to $d$-dimension using two additional linear layers ($P_l, P_l^a$):

$$z \leftarrow zP_l \in \mathbb{R}^{1 \times d}, A_l = AP_l^a \in \mathbb{R}^{N \times d}. \tag{3}$$

The linguistic feature $z$ is then encoded into the dual-space using cross-attention:

$$z' = \text{Attn}(z, A_l, A_l), f_l = z' + z \in \mathbb{R}^{1 \times d}, g_l = f_l A_l^T \in \mathbb{R}^{1 \times N}, \tag{4}$$

It is notable that only cross-attention is employed to encode the linguistic feature, as overly intricate operations on the textual side may lead to overfitting.

**Multi-Modality Alignment.** We use the contrastive loss [48] to align the visual-linguistic features within the dual-space. Specifically, the contrastive loss in the latent space is defined as:

$$L_{cl}(f_v, f_l) = -\log \frac{\exp(f_v \cdot f_l^T/\tau)}{\sum_j \exp(f_v \cdot f_{l,j}^T/\tau)} - \log \frac{\exp(f_l \cdot f_v^T/\tau)}{\sum_j \exp(f_l \cdot f_{v,j}^T/\tau)}, \tag{5}$$

where $f_{v,j}$ and $f_{l,j}$ are the $\ell 2$-normalized features of the $j$-th input image and text, respectively. $\tau$ is a temperature hyperparameter. The contrastive loss in the attribute space is formulated as follows:

$$L_{ca}(g_v, g_l) = -\log \frac{\exp(g_v \cdot g_l^T/\tau)}{\sum_j \exp(g_v \cdot g_{l,j}^T/\tau)} - \log \frac{\exp(g_l \cdot g_v^T/\tau)}{\sum_j \exp(g_l \cdot g_{v,j}^T/\tau)}, \tag{6}$$

where $g_{v,j}$ and $g_{l,j}$ denote the $\ell 2$-normalized visual and linguistic attribute representations of the $j$-th input image and text, respectively.

### 3.2 Feature Semantic Topology Preservation

Compared with conventional ZSL, the GZSL problem is more challenging: the model needs to decide between both seen and unseen classes without knowing whether the test example comes from seen or unseen classes. Previous GZSL methods [5, 49] observed the domain bias problem: *trained model is seriously biased towards seen classes in the testing phase*. The bias arises because the model is only trained on seen classes and, therefore, learns features and patterns specific to those classes. This phenomenon often leads to inferior performance on unseen classes in the GZSL evaluation setting. For VLM methods such as prompt learning [8, 10, 11], although the model weights are frozen, another problem called weak generalizability problem is observed [8]: *the learned prompt is not generalizable to unseen classes within the same dataset*. A reasonable explanation is the learned task-specific prompt overfits the seen base classes when finetuing VLMs. CoCoOP [8] try to alleviate

this problem by using instance-conditioned prompt, but the evaluation is only in the base-to-novel setting rather than the more challenging GZSL setting [4]. To our knowledge, in the VLM context, no prior work has evaluated the challenging GZSL setting, let alone a principled solution to tackle the generalization problem under this setting.

In this paper, we tackle the weak generalizability problem in the feature space, particularly our newly introduced attribute space. Intuitively, the design of base attribute vocabulary has the effect of combating overfitting to some extent. But here, we are looking for a more effective and principled solution. A straightforward idea is to avoid overfitting to the seen classes by regularizing the features during the model finetuning process. For example, we can constrain the rank of image features to be no-decreasing, which boils down to maximizing the nuclear norm of the feature matrix [50]. However, this method empirically proves less effective. Then, we observe that the pre-trained VLMs such as CLIP perform equally well on both seen and unseen classes (Fig. 1 and also [8, 10, 11]). This motivates us to inherit the generalization capability of CLIP to prevent overfitting to seen classes during the finetuning process, as shown in Fig. 1(b-d). Specifically, we want to maintain the class topology of CLIP embedding space as a way for generalization inheritance. The class topology is composed of all pairwise class angles, which can be calculated by cosine similarities (CLIP embedding is $\ell$-2 normalized). Denote the textual description features of all $c$ (seen+unseen) classes in CLIP's embedding space as $Z = \{z^1, z^2, \ldots, z^c\} \in \mathbb{R}^{c \times d}$. The corresponding features in the attribute space are denoted as $G_l = \{g_l^1, g_l^2, \ldots, g_l^c\} \in \mathbb{R}^{c \times N}$. Finally, in order to preserve the class topology in attribute space to be similar to that in CLIP embedding space, we adopt the Pearson correlation coefficient to define a topology-preserving loss as follows:

$$L_{tp}(Z, G_l) = -\frac{\sum_{i,j}^c (w_{ij} - \frac{1}{c^2}\sum_{ij}^c w_{ij})(\tilde{w}_{ij} - \frac{1}{c^2}\sum_{ij}^c \tilde{w}_{ij})}{\sqrt{\sum_{ij}^c \left(w_{ij} - \frac{1}{c^2}\sum_{ij}^c w_{ij}\right)^2}\sqrt{\sum_{ij}^c \left(\tilde{w}_{ij} - \frac{1}{c^2}\sum_{ij}^c \tilde{w}_{ij}\right)^2}}, \tag{7}$$

where $w_{ij} = \frac{z^i \cdot z^{j^T}}{||z^i||_2 ||z^j||_2}, \tilde{w}_{ij} = \frac{g_l^i \cdot g_l^{j^T}}{||g_l^i||_2 ||g_l^j||_2}, i, j = 1, 2, .., c$. Note that Eq. 7 maintains the pairwise feature angles between categories before and after finetuning. Hence, both seen and unseen classes can leverage the class topology of CLIP for better GZSL classification. More importantly, with $L_{tp}(Z, G_l)$, the finetuned model is less likely to be seriously biased towards seen classes, thus improving the generalization to unseen classes.

### 3.3 Learning Objective and Inference

The overall learning objective is:

$$L(\theta) = L_{cl} + \eta L_{ca} + \beta L_{tp}, \tag{8}$$

where $\eta$ and $\beta$ are loss weights, and $\theta$ denotes model's trainable parameters. During inference, given a testing sample, we first extract its visual feature $f_v$ and perform the nearest neighbor search from all seen and unseen classes:

$$\underset{y \in \mathcal{Y}^s \cup \mathcal{Y}^u}{\arg\max} \ f_v \cdot f_{l,y}^T, \tag{9}$$

where $f_{l,y}$ is the linguistic feature of class $y$ (all vectors are $\ell$2-normalized).

## 4 Experiments

**Datasets and Metrics.** We validate the effectiveness of TPR on four widely-used datasets in GZSL: AwA2 [51], CUB [42], FLO [52], and SUN [53]. We follow the commonly-used dataset split [26] but use the generated textual descriptions instead of attribute annotation. To further evaluate the generalization ability, we conduct experiments on eight other object recognition datasets: FGVC-Aircraft [54], Country [9], StanfordCars [55], EuroSAT [56], DTD [57], UCF101 [58], Food101 [59], and OxfordPets [60]. These datasets are divided into seen and unseen classes in a similar way. Note that under the GZSL setting, the model can only be trained on seen classes and evaluated on both seen and unseen classes to evaluate its generalization ability. We report the average per-class top-1 accuracy on *seen classes* ($S$) and *unseen classes* ($U$), respectively. To balance the two metrics, we also report the *harmonic mean* ($H$) of the seen and unseen accuracy: $H = 2 \times \frac{S \times U}{S + U}$.

Table 1: Comparison with state-of-the-art methods in generalized zero-shot learning (GZSL) setting. The proposed TPR obtains the best harmonic mean ($H$) in 11 out of 12 datasets. Among them, TPR performs the best on all fine-grained datasets (marked with *), and best (**bolded**) or second best (underlined) on almost all other datasets. Besides CLIP, TPR also works well with other VLMs: †CoCa [36] (ViT-B/32) and ‡EVA-02 [62] (ViT-B/16). Equipped with the latter, we can further push forward existing SOTA by a large margin.

| Model | AwA2 | | | CUB* | | | FLO* | | | SUN | | | FGVC-Aircraft* | | | Country | | |
|---|---|---|---|---|---|---|---|---|---|---|---|---|---|---|---|---|---|---|
| | S | U | H | S | U | H | S | U | H | S | U | H | S | U | H | S | U | H |
| CLIP [9] | 81.69 | 77.66 | 79.62 | 29.88 | 29.61 | 29.74 | 53.91 | 51.16 | 52.50 | 46.28 | 49.51 | 47.84 | 18.25 | 11.15 | 13.84 | 13.16 | 12.13 | 12.62 |
| CoOp [40] | 81.36 | 69.42 | 74.92 | 22.23 | 18.23 | 20.03 | 56.27 | 50.65 | 53.31 | 49.85 | 49.31 | 49.57 | 17.13 | 12.10 | 14.18 | 12.86 | 9.73 | 11.08 |
| CoCoOp [8] | 78.53 | 73.81 | 76.10 | 23.53 | 19.81 | 21.51 | 60.21 | 50.22 | 54.76 | 49.53 | 49.51 | 49.52 | 18.81 | 13.60 | 15.79 | 13.59 | 8.03 | 10.09 |
| MaPLe [10] | 78.04 | 71.25 | 74.49 | 22.46 | 20.66 | 21.52 | 59.88 | 48.39 | 53.52 | 46.82 | 48.68 | 47.73 | 21.75 | 15.20 | 17.89 | 12.96 | 9.54 | 10.99 |
| PromptSRC [11] | 84.04 | 70.73 | 76.82 | 30.92 | 16.32 | 21.37 | 60.68 | 54.45 | 57.40 | 47.83 | 49.24 | 48.52 | 23.44 | 13.10 | 16.81 | 14.42 | 6.87 | 9.30 |
| ProGrad [12] | 81.73 | 67.46 | 73.91 | 22.97 | 21.38 | 22.15 | 61.21 | 50.53 | 55.36 | 52.71 | 49.44 | 51.03 | 19.00 | 11.00 | 13.93 | 13.99 | 8.77 | 10.78 |
| CE [5] | 76.69 | 67.80 | 71.97 | 31.80 | 19.01 | 23.80 | 63.02 | 44.09 | 51.88 | 44.11 | 47.15 | 45.58 | 28.63 | 25.25 | 26.83 | 12.80 | 8.07 | 9.90 |
| LSA [6] | 77.16 | 65.87 | 71.07 | 37.35 | 19.54 | 25.66 | 77.51 | 41.03 | 53.66 | 45.66 | 48.19 | 46.89 | 29.44 | 27.85 | 28.62 | 12.21 | 7.51 | 9.30 |
| ZLAP [7] | 76.35 | 74.74 | 75.54 | 32.41 | 25.51 | 28.55 | 68.22 | 54.77 | 60.76 | 48.18 | 47.29 | 47.73 | 29.38 | 27.10 | 28.19 | 12.64 | 10.42 | 11.32 |
| TPR | 87.10 | 76.81 | 81.63 | 41.22 | 26.87 | 32.53 | 77.58 | 64.52 | 70.45 | 50.47 | 45.40 | 47.80 | 36.88 | 29.65 | 32.87 | 18.75 | 16.03 | 17.28 |
| TPR† | 80.52 | 71.70 | 75.86 | 42.42 | 25.97 | 32.22 | 82.62 | 62.99 | 71.48 | 50.08 | 45.49 | 47.67 | 34.63 | 31.25 | 32.85 | 20.18 | 15.68 | 17.65 |
| TPR‡ | 95.60 | 78.81 | 86.39 | 53.10 | 32.55 | 40.36 | 83.75 | 64.65 | 72.97 | 58.29 | 52.08 | 55.01 | 43.50 | 31.30 | 36.41 | 27.82 | 23.31 | 25.37 |

| Model | StanfordCars* | | | EuroSAT | | | DTD | | | UCF101* | | | Food101* | | | OxfordPets* | | |
|---|---|---|---|---|---|---|---|---|---|---|---|---|---|---|---|---|---|---|
| | S | U | H | S | U | H | S | U | H | S | U | H | S | U | H | S | U | H |
| CLIP [9] | 46.65 | 37.78 | 41.75 | 21.13 | 11.25 | 14.68 | 36.39 | 41.39 | 38.73 | 53.72 | 64.92 | 58.79 | 67.74 | 73.05 | 70.29 | 82.67 | 65.83 | 73.29 |
| CoOp [40] | 49.86 | 38.47 | 43.43 | 29.89 | 12.27 | 17.40 | 44.34 | 36.56 | 40.07 | 62.13 | 47.41 | 53.78 | 71.82 | 64.64 | 68.04 | 73.47 | 57.66 | 64.61 |
| CoCoOp [8] | 51.93 | 37.84 | 43.78 | 52.64 | 18.34 | 27.21 | 42.19 | 35.94 | 38.82 | 58.29 | 60.62 | 59.43 | 72.55 | 60.42 | 65.93 | 72.53 | 58.97 | 65.05 |
| MaPLe [10] | 55.29 | 35.67 | 43.36 | 30.72 | 19.52 | 23.87 | 42.25 | 39.72 | 40.95 | 55.37 | 62.51 | 58.73 | 72.16 | 71.47 | 71.82 | 75.87 | 56.01 | 64.45 |
| PromptSRC [11] | 55.56 | 39.85 | 46.41 | 28.71 | 14.40 | 19.18 | 51.30 | 42.56 | 46.52 | 61.92 | 59.89 | 60.89 | 77.06 | 56.31 | 65.07 | 78.60 | 52.99 | 63.30 |
| ProGrad [12] | 52.20 | 35.36 | 42.16 | 59.14 | 17.12 | 26.55 | 54.62 | 39.78 | 46.03 | 60.10 | 58.76 | 59.42 | 73.48 | 64.26 | 68.56 | 72.53 | 59.95 | 65.64 |
| CE [5] | 56.62 | 40.94 | 47.52 | 61.61 | 32.88 | 42.88 | 44.79 | 29.61 | 35.65 | 54.78 | 34.66 | 42.46 | 70.48 | 53.88 | 61.07 | 71.07 | 59.54 | 64.80 |
| LSA [6] | 59.19 | 41.41 | 48.73 | 55.10 | 24.89 | 34.29 | 45.64 | 27.72 | 34.49 | 51.76 | 37.22 | 43.30 | 69.10 | 53.82 | 60.51 | 73.73 | 59.27 | 65.71 |
| ZLAP [7] | 49.60 | 43.22 | 46.19 | 74.59 | 23.22 | 35.41 | 46.94 | 30.94 | 37.30 | 56.89 | 42.65 | 48.75 | 70.20 | 60.86 | 65.20 | 72.93 | 59.89 | 65.77 |
| TPR | 69.48 | 46.33 | 55.59 | 82.78 | 45.73 | 58.91 | 55.47 | 42.06 | 47.84 | 69.14 | 66.88 | 67.99 | 93.67 | 85.41 | 89.35 | 90.60 | 66.39 | 76.63 |
| TPR† | 88.09 | 70.84 | 78.53 | 75.60 | 61.43 | 67.78 | 62.70 | 45.39 | 52.66 | 74.20 | 61.81 | 67.44 | 82.37 | 74.22 | 78.08 | 84.07 | 71.32 | 77.17 |
| TPR‡ | 87.25 | 73.57 | 79.83 | 76.07 | 56.41 | 64.78 | 68.95 | 46.39 | 55.46 | 75.01 | 74.57 | 74.79 | 88.87 | 79.97 | 84.18 | 92.27 | 70.17 | 79.72 |

**Textual Description Generation.** We propose employing ChatGPT [61] to autonomously generate a descriptive paragraph for each class, leveraging its extensive knowledge base learned from diverse sources in the Internet. To facilitate this process for each dataset, we advocate employing the following prompt structure: "*I have never seen images of <type>. The following is a good description of <class name>, so I can easily recognize <class name>.*" Here, the placeholder <type> corresponds to categories such as animals, birds, flowers, *etc.*, while <class name> denotes the specific name of each class. As an illustration, consider the antelope class from the AwA2 dataset, where the generated description reads as follows: *"Antelopes are herbivorous mammals known for their slender bodies and long, curved horns. They are often found in grasslands and open savannas, where they graze on vegetation and use their speed and agility to escape predators."*

**Training Details.** We employ pretrained ViT-B/32 CLIP as the feature extractor, which outputs 512-dimensional visual and linguistic features. In total, we collect $N_1 = 5,996$ attribute words and we use pretrained Bert [46] to extract attribute features of dimension $d_a = 768$. For all datasets, TPR is trained for 200 epochs with a batch-size of 512 via Adam optimizer on a single NVIDIA RTX4090 GPU. Overall, optimal hyperparameters of TPR are chosen from the following ranges: learning rate $\in \{$1e-5, 3e-5, 5e-5, 7e-5, 1e-4$\}$, $\eta \in \{$0.2, 0.5, 1.0, 2.0$\}$, $\beta \in \{$1e-4, 5e-4, 1e-3$\}$, $\tau \in \{$0.03, 0.05, 0.07, 0.10$\}$, and $N_2 \in \{100, 200, 300, 400\}$, which are tuned on the validation set via grid search.

## 4.1 Comparison with the SoTA

We compare with the state-of-the-art methods including prompt learning and generative-based GZSL. We adapt them to our description-based GZSL setting for a fair comparison. (1) *Prompt learning methods*, including CoOp [40], CoCoOp [8], MaPLe [10], PromptSRC [11] and ProGrad [12], fine-tune the pretrained CLIP through integrating trainable prompt tokens within a predefined template. To adapt these methods for GZSL, we input (image, label) pairs while omitting the utilization of textual descriptions. Notably, our setting differs from the base-to-novel setting [40] typically used in these methods, which originally predicts the seen and unseen classes **separately**. Instead, we conduct the comparison in the GZSL setting [26] where the prediction space includes **both** the seen and unseen classes, thereby posing a *more challenging task than the base-to-novel setting*. (2) *Generative-based GZSL methods*, including CE [5], LSA [6] and ZLAP [7], are adapted by substituting the input semantic features derived from attribute annotations with those obtained from textual descriptions. For a fair comparison, all methods adopt the same backbone and dataset split.

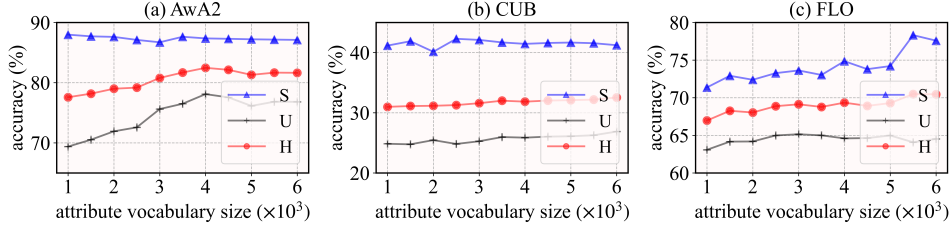

Figure 3: Impact of the size of the attribute vocabulary (*i.e.*, $N_1$) on model performance.

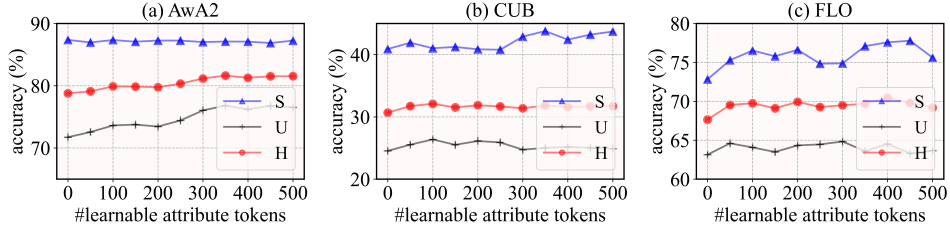

Figure 4: Impact of the number of learnable attribute tokens (*i.e.*, $N_2$) on model performance.

The performance analysis presented in Table 1 reveals that TPR outperforms other methods in terms of $H$ metric in 11 out of 12 datasets. Compared with prompt learning methods, TPR obtains an average absolute performance improvement of 12.46%, 9.46%, and 11.94% on $S$, $U$ and $H$ metrics, respectively. Similarly, in comparison with generative-based GZSL methods, TPR demonstrates an average absolute performance gain of 11.24%, 9.29%, and 10.68% on $S$, $U$ and $H$ metrics, respectively. Remarkably, TPR achieves the best performance on all fine-grained datasets (marked with *), underscoring its exceptional fine-grained data perception capability. Furthermore, while the recognition performance of many baselines, such as CoCoOp, deteriorates for unseen classes compared to zero-shot CLIP, TPR continues to achieve outstanding results. This indicates TPR's proficiency to effectively preserve CLIP's generalization capability to unseen classes. To verify the applicability of TPR to different VLMs, we utilize two VLMs, CoCa [36] and EVA-02[62], as backbone networks for feature extraction from input images and text descriptions. The results, presented at the bottom of Table 1, demonstrate TPR's robust generalization across various VLMs.

## 4.2 Ablation Study

**Loss Functions.** We study the efficacy of two novel module designs in Table 2. By augmenting the latent space with the attribute space ($L_{cl} \rightarrow L_{cl} + L_{ca}$), the accuracy of unseen classes is improved by 4.26%, 2.78%, and 3.81% for the three datasets. This underscores the attribute space's adeptness in capturing attribute features overlooked by the latent space, thus promoting the perception of unseen classes. Meanwhile, the inclusion of topology-preserving objective ($L_{cl} + L_{tp}$) results in a notable accuracy improvement for unseen classes, manifesting as an increase of 6.36%, 2.91%, and 1.81% across the three datasets. Finally, incorporating both objectives ($L_{ca}$ and $L_{tp}$) into our full model, the accuracy outperforms each single objective on almost all metrics (seen, unseen and harmonic) except for seen on CUB, demonstrating the complementary effect of two devised modules: dual-space feature alignment ($L_{ca}$) and topology-preserving objective ($L_{tp}$).

**Attribute Reservoir.** We conduct an ablation study across various configurations of the attribute reservoir, as presented in Table 2. Notably, competitive performance is attained when solely relying on the static attribute vocabulary. Conversely, utilizing solely learnable attribute tokens exhibits improved performance on seen classes, albeit with a decrease in accuracy for recognizing unseen classes. Optimal $H$ performance is achieved when both types of attribute knowledge are combined.

**Static Vocabulary Size** $N_1$. In general, a larger $N_1$ entails the inclusion of a greater number of attribute words, facilitating a finer-grained representation of objects and potentially enhancing recognition performance. The findings depicted in Fig. 3 substantiate this notion. It is evident that as $N_1$ increases, the model performance improves significantly, especially for unseen classes. Overall, the harmonic mean $H$ tends to saturate beyond 4,000 attribute words on the coarse-grained AwA2

Table 2: Ablation on different regularizers and reservoir components.

| Setting | | AwA2 | | | CUB | | | FLO | | |
|---|---|---|---|---|---|---|---|---|---|---|
| | | $S$ | $U$ | $H$ | $S$ | $U$ | $H$ | $S$ | $U$ | $H$ |
| full model | | 87.10 | **76.81** | **81.63** | 41.22 | **26.87** | **32.53** | 77.58 | **64.52** | 70.45 |
| objective | $L_{cl}$ | 85.70 | 68.14 | 75.92 | **41.55** | 23.24 | 29.81 | 74.01 | 60.45 | 66.55 |
| | $L_{cl}+L_{ca}$ | 84.69 | 72.40 | 78.06 | 40.77 | 26.02 | 31.77 | 74.41 | 64.26 | 68.96 |
| | $L_{cl}+L_{tp}$ | 84.44 | 74.50 | 79.16 | 40.72 | 26.15 | 31.85 | 76.72 | 62.26 | 68.74 |
| reservoir | static vocabulary | 86.71 | 75.23 | 80.56 | 41.11 | 26.12 | 31.94 | 74.62 | 64.11 | 68.97 |
| | learnable tokens | **87.69** | 64.58 | 74.38 | 41.27 | 20.85 | 27.70 | **82.72** | 60.71 | 70.03 |

Table 3: Ablation on the topology preserving loss $L_{tp}$.

| Setting | AwA2 | | | CUB | | | FLO | | |
|---|---|---|---|---|---|---|---|---|---|
| | $S$ | $U$ | $H$ | $S$ | $U$ | $H$ | $S$ | $U$ | $H$ |
| w/o $L_{tp}$ | 84.69 | 72.40 | 78.06 | 40.77 | 26.02 | 31.77 | 74.41 | 64.26 | 68.96 |
| nuclear norm | **87.88** | 72.48 | 79.44 | 41.09 | 25.36 | 31.36 | 74.72 | 64.47 | 69.22 |
| orthogonality | 87.15 | 74.87 | 80.55 | 40.36 | 26.13 | 31.72 | 73.68 | **64.95** | 69.04 |
| $L_{tp}^{lat}$ | 86.93 | **77.00** | **81.67** | 40.61 | 26.00 | 31.70 | 74.65 | 64.34 | 69.11 |
| $L_{tp}$ | 87.10 | 76.81 | 81.63 | **41.22** | **26.87** | **32.53** | 77.58 | 64.52 | **70.45** |

Table 4: Comparison with the state-of-the-art prompt learning methods under *base-to-novel* setting [8].

| Method | EuroSAT | | | DTD | | | Food101 | | |
|---|---|---|---|---|---|---|---|---|---|
| | Base | Novel | HM | Base | Novel | HM | Base | Novel | HM |
| CLIP [9] | 56.48 | 64.05 | 60.03 | 53.24 | 59.90 | 56.37 | 90.10 | 91.22 | 90.66 |
| CoOp [40] | 92.19 | 54.74 | 68.69 | 79.44 | 41.18 | 54.24 | 88.33 | 82.26 | 85.19 |
| CoCoOp [8] | 87.49 | 60.04 | 71.21 | 77.01 | 56.00 | 64.85 | 90.70 | 91.29 | 90.99 |
| MaPLe [10] | 94.07 | 73.23 | 82.35 | 80.36 | 59.18 | 68.16 | 90.71 | **92.05** | 91.38 |
| PromptSRC [11] | 92.90 | 73.90 | 82.32 | 83.37 | **62.97** | **71.75** | 90.67 | 91.53 | 91.10 |
| TPR | **95.12** | **76.66** | **84.90** | **84.80** | 59.39 | 69.86 | **94.03** | 91.15 | **92.57** |

dataset, while $H$ continues to increase on the fine-grained CUB and FLO datasets. This again verifies the effectiveness of our model in capturing the fine-grained complex patterns.

**Learnable Tokens Quantity** $N_2$. As shown in Fig. 4, the accuracy of unseen classes on the AwA2 dataset exhibits improvement with increasing $N_2$. However, for the CUB and FLO datasets, the accuracy of unseen classes tends to plateau around 200 tokens, diverging from the pattern on AwA2. We conjecture that an excessive number of learnable tokens may lead to overfitting on seen classes for fine-grained datasets, thereby limiting further enhancement.

**Topology-Preserving Constraint.** We ablate on various choices of the topology-preserving loss. These variants encompass: (1) *nuclear norm*: maximizing the nuclear norm to ensure that visual features span the entire space; (2) *orthogonality*: enforcing orthogonality among class text features in the attribute space; (3) *topology-preservation in latent space* ($L_{tp}^{lat}$): maintaining pairwise angles between class features (Eq. 7) in the latent space; (4) *our topology preservation in attribute space* ($L_{tp}$). From Table 3, we observe that both nuclear norm and orthogonality cannot consistently improve the performance, possibly due to the lack of reference on CLIP. As to the topology-preserving objective, enforcing it on the novel attribute space yields consistently superior performance, which again reveals the better expressivity of the proposed attribute space.

**Base-to-Novel Evaluation.** As the prompt learning methods originally evaluate on the base-to-novel setting, we compare with them on standard prompt learning benchmarks for completeness and better understanding of the GZSL setting. Comparing Table 1 and Table 4, we notice that the performance of all methods in base-to-novel setting surpasses the corresponding GZSL results, corroborating that GZSL is a more challenging task. Moreover, our method outperforms or at least on-par-with the prompt learning methods in the base-to-novel setting.

## 5 Conclusion and Future Work

In this paper, we tackle the challenging GZSL problem by designing a Topology-Preserving Reservoir (TPR) model. Specifically, TPR embraces two unique designs for VLMs: a dual-space feature alignment module and a feature semantic topology preserving objective. First, a reservoir containing

both static and learnable vocabulary tokens is devised to construct a representative attribute space to enhance the latent space, which facilitates the exploitation of complex and fine-grained visual-linguistic patterns. Second, we propose a topology-preserving objective, which inherits the good generalization ability of CLIP to mitigate the weak generalization problem of prompt learning methods. In particular, topology-preserving objective constrains the variations of angles between pairwise categories before and after CLIP finetuning. Comprehensive experiments are conducted on twelve object recognition datasets, validating the superior performance of our method in the challenging and more practical GZSL setting. In the future, we want to investigate TPR on other applications such as few-shot learning and other modalities such as video.

**Limitations**. Text alone may not fully capture the nuances of fine-grained datasets like CUB, while attribute annotations, though more accurate, are costly. Thus, a more desirable solution would be combining the knowledge from expert-provided attribute annotations with LLM-generated text to enhance performance. Additionally, our method may face challenges in aligning visual features of generic scenes with description features in the attribute space, especially when descriptions are not sufficiently specific. This could be alleviated by providing more distinct and human-refined descriptions.

## Acknowledgments

We thank all the anonymous reviewers and ACs for their valuable comments. This work was supported by the National Science Foundation of China (No. 62088102).

## Footnotes

[3]For unseen, we only use the class names without accessing any additional data.

[4]In Table 1, we evaluate CoCoOP under GZSL setting, but the performance is unsatisfying on unseen classes.

## References

[1] Christoph H Lampert, Hannes Nickisch, and Stefan Harmeling. Attribute-based classification for zero-shot visual object categorization. *IEEE transactions on pattern analysis and machine intelligence*, 36(3):453–465, 2013.

[2] Mohamed Elhoseiny, Babak Saleh, and Ahmed Elgammal. Write a classifier: Zero-shot learning using purely textual descriptions. In *Proceedings of the IEEE International Conference on Computer Vision*, pages 2584–2591, 2013.

[3] Walter J Scheirer, Anderson de Rezende Rocha, Archana Sapkota, and Terrance E Boult. Toward open set recognition. *IEEE transactions on pattern analysis and machine intelligence*, 35(7):1757–1772, 2012.

[4] Yongqin Xian, Christoph H Lampert, Bernt Schiele, and Zeynep Akata. Zero-shot learning—a comprehensive evaluation of the good, the bad and the ugly. *IEEE transactions on pattern analysis and machine intelligence*, 41(9):2251–2265, 2018.

[5] Zongyan Han, Zhenyong Fu, Shuo Chen, and Jian Yang. Contrastive embedding for generalized zero-shot learning. In *Proceedings of the IEEE/CVF conference on computer vision and pattern recognition*, pages 2371–2381, 2021.

[6] Celina Hanouti and Hervé Le Borgne. Learning semantic ambiguities for zero-shot learning. *Multimedia Tools and Applications*, pages 1–15, 2023.

[7] Dubing Chen, Yuming Shen, Haofeng Zhang, and Philip H.S. Torr. Zero-shot logit adjustment. In Lud De Raedt, editor, *Proceedings of the Thirty-First International Joint Conference on Artificial Intelligence, IJCAI-22*, pages 813–819. International Joint Conferences on Artificial Intelligence Organization, 7 2022. Main Track.

[8] Kaiyang Zhou, Jingkang Yang, Chen Change Loy, and Ziwei Liu. Conditional prompt learning for vision-language models. In *Proceedings of the IEEE/CVF Conference on Computer Vision and Pattern Recognition*, pages 16816–16825, 2022.

[9] Alec Radford, Jong Wook Kim, Chris Hallacy, Aditya Ramesh, Gabriel Goh, Sandhini Agarwal, Girish Sastry, Amanda Askell, Pamela Mishkin, Jack Clark, et al. Learning transferable visual models from natural language supervision. In *International conference on machine learning*, pages 8748–8763. PMLR, 2021.

[10] Muhammad Uzair Khattak, Hanoona Rasheed, Muhammad Maaz, Salman Khan, and Fahad Shahbaz Khan. Maple: Multi-modal prompt learning. In *Proceedings of the IEEE/CVF Conference on Computer Vision and Pattern Recognition*, pages 19113–19122, 2023.

[11] Muhammad Uzair Khattak, Syed Talal Wasim, Muzammal Naseer, Salman Khan, Ming-Hsuan Yang, and Fahad Shahbaz Khan. Self-regulating prompts: Foundational model adaptation without forgetting. In *Proceedings of the IEEE/CVF International Conference on Computer Vision*, pages 15190–15200, 2023.

[12] Beier Zhu, Yulei Niu, Yucheng Han, Yue Wu, and Hanwang Zhang. Prompt-aligned gradient for prompt tuning. In *Proceedings of the IEEE/CVF International Conference on Computer Vision*, pages 15659–15669, 2023.

[13] Jimmy Lei Ba, Kevin Swersky, Sanja Fidler, et al. Predicting deep zero-shot convolutional neural networks using textual descriptions. In *Proceedings of the IEEE international conference on computer vision*, pages 4247–4255, 2015.

[14] Wei-Lun Chao, Soravit Changpinyo, Boqing Gong, and Fei Sha. An empirical study and analysis of generalized zero-shot learning for object recognition in the wild. In *Computer Vision–ECCV 2016: 14th European Conference, Amsterdam, The Netherlands, October 11-14, 2016, Proceedings, Part II 14*, pages 52–68. Springer, 2016.

[15] Soravit Changpinyo, Wei-Lun Chao, Boqing Gong, and Fei Sha. Synthesized classifiers for zero-shot learning. In *Proceedings of the IEEE conference on computer vision and pattern recognition*, pages 5327–5336, 2016.

[16] Shichen Liu, Mingsheng Long, Jianmin Wang, and Michael I Jordan. Generalized zero-shot learning with deep calibration network. *Advances in neural information processing systems*, 31, 2018.

[17] Man Liu, Feng Li, Chunjie Zhang, Yunchao Wei, Huihui Bai, and Yao Zhao. Progressive semantic-visual mutual adaption for generalized zero-shot learning. In *Proceedings of the IEEE/CVF Conference on Computer Vision and Pattern Recognition*, pages 15337–15346, 2023.

[18] Vinay Verma, Nikhil Mehta, Kevin J Liang, Aakansha Mishra, and Lawrence Carin. Meta-learned attribute self-interaction network for continual and generalized zero-shot learning. In *Proceedings of the IEEE/CVF Winter Conference on Applications of Computer Vision*, pages 2721–2731, 2024.

[19] Guo-Sen Xie, Li Liu, Fan Zhu, Fang Zhao, Zheng Zhang, Yazhou Yao, Jie Qin, and Ling Shao. Region graph embedding network for zero-shot learning. In *Computer Vision–ECCV 2020: 16th European Conference, Glasgow, UK, August 23–28, 2020, Proceedings, Part IV 16*, pages 562–580. Springer, 2020.

[20] Mert Bulent Sariyildiz and Ramazan Gokberk Cinbis. Gradient matching generative networks for zero-shot learning. In *Proceedings of the IEEE/CVF conference on computer vision and pattern recognition*, pages 2168–2178, 2019.

[21] Wenjia Xu, Yongqin Xian, Jiuniu Wang, Bernt Schiele, and Zeynep Akata. Attribute prototype network for zero-shot learning. *Advances in Neural Information Processing Systems*, 33:21969–21980, 2020.

[22] Xiaolong Wang, Yufei Ye, and Abhinav Gupta. Zero-shot recognition via semantic embeddings and knowledge graphs. In *Proceedings of the IEEE conference on computer vision and pattern recognition*, pages 6857–6866, 2018.

[23] Andrea Frome, Greg S Corrado, Jon Shlens, Samy Bengio, Jeff Dean, Marc'Aurelio Ranzato, and Tomas Mikolov. Devise: A deep visual-semantic embedding model. *Advances in neural information processing systems*, 26, 2013.

[24] Muhammad Ferjad Naeem, Yongqin Xian, Luc V Gool, and Federico Tombari. I2dformer: Learning image to document attention for zero-shot image classification. *Advances in Neural Information Processing Systems*, 35:12283–12294, 2022.

[25] He Huang, Changhu Wang, Philip S Yu, and Chang-Dong Wang. Generative dual adversarial network for generalized zero-shot learning. In *Proceedings of the IEEE/CVF conference on computer vision and pattern recognition*, pages 801–810, 2019.

[26] Yongqin Xian, Tobias Lorenz, Bernt Schiele, and Zeynep Akata. Feature generating networks for zero-shot learning. In *Proceedings of the IEEE conference on computer vision and pattern recognition*, pages 5542–5551, 2018.

[27] Yongqin Xian, Saurabh Sharma, Bernt Schiele, and Zeynep Akata. f-vaegan-d2: A feature generating framework for any-shot learning. In *Proceedings of the IEEE/CVF conference on computer vision and pattern recognition*, pages 10275–10284, 2019.

[28] Junhan Kim, Kyuhong Shim, and Byonghyo Shim. Semantic feature extraction for generalized zero-shot learning. In *Proceedings of the AAAI Conference on Artificial Intelligence*, volume 36, pages 1166–1173, 2022.

[29] Muhammad Ferjad Naeem, Muhammad Gul Zain Ali Khan, Yongqin Xian, Muhammad Zeshan Afzal, Didier Stricker, Luc Van Gool, and Federico Tombari. I2mvformer: Large language model generated multi-view document supervision for zero-shot image classification. In *Proceedings of the IEEE/CVF Conference on Computer Vision and Pattern Recognition*, pages 15169–15179, 2023.

[30] Yassine Ouali, Adrian Bulat, Brais Matinez, and Georgios Tzimiropoulos. Black box few-shot adaptation for vision-language models. In *Proceedings of the IEEE/CVF International Conference on Computer Vision*, pages 15534–15546, 2023.

[31] Hui Chen, Jingjing Jiang, and Nanning Zheng. Learning to infer unseen single-/multi-attribute-object compositions with graph networks. *IEEE Transactions on Pattern Analysis and Machine Intelligence*, 2023.

[32] Ting Chen, Simon Kornblith, Mohammad Norouzi, and Geoffrey Hinton. A simple framework for contrastive learning of visual representations. In *International conference on machine learning*, pages 1597–1607. PMLR, 2020.

[33] Chao Jia, Yinfei Yang, Ye Xia, Yi-Ting Chen, Zarana Parekh, Hieu Pham, Quoc Le, Yun-Hsuan Sung, Zhen Li, and Tom Duerig. Scaling up visual and vision-language representation learning with noisy text supervision. In *International conference on machine learning*, pages 4904–4916. PMLR, 2021.

[34] Junnan Li, Dongxu Li, Caiming Xiong, and Steven Hoi. Blip: Bootstrapping language-image pre-training for unified vision-language understanding and generation. In *International Conference on Machine Learning*, pages 12888–12900. PMLR, 2022.

[35] Hangbo Bao, Wenhui Wang, Li Dong, Qiang Liu, Owais Khan Mohammed, Kriti Aggarwal, Subhojit Som, Songhao Piao, and Furu Wei. Vlmo: Unified vision-language pre-training with mixture-of-modality-experts. *Advances in Neural Information Processing Systems*, 35:32897–32912, 2022.

[36] Jiahui Yu, Zirui Wang, Vijay Vasudevan, Legg Yeung, Mojtaba Seyedhosseini, and Yonghui Wu. Coca: Contrastive captioners are image-text foundation models. *arXiv preprint arXiv:2205.01917*, 2022.

[37] Yanghao Li, Haoqi Fan, Ronghang Hu, Christoph Feichtenhofer, and Kaiming He. Scaling language-image pre-training via masking. In *Proceedings of the IEEE/CVF Conference on Computer Vision and Pattern Recognition*, pages 23390–23400, 2023.

[38] Shengnan An, Yifei Li, Zeqi Lin, Qian Liu, Bei Chen, Qiang Fu, Weizhu Chen, Nanning Zheng, and Jian-Guang Lou. Input-tuning: Adapting unfamiliar inputs to frozen pretrained models. *arXiv preprint arXiv:2203.03131*, 2022.

[39] Pengfei Liu, Weizhe Yuan, Jinlan Fu, Zhengbao Jiang, Hiroaki Hayashi, and Graham Neubig. Pre-train, prompt, and predict: A systematic survey of prompting methods in natural language processing. *ACM Computing Surveys*, 55(9):1–35, 2023.

[40] Kaiyang Zhou, Jingkang Yang, Chen Change Loy, and Ziwei Liu. Learning to prompt for vision-language models. *International Journal of Computer Vision*, 130(9):2337–2348, 2022.

[41] Yabin Wang, Zhiwu Huang, and Xiaopeng Hong. S-prompts learning with pre-trained transformers: An occam's razor for domain incremental learning. *Advances in Neural Information Processing Systems*, 35:5682–5695, 2022.

[42] Catherine Wah, Steve Branson, Peter Welinder, Pietro Perona, and Serge Belongie. The caltech-ucsd birds-200-2011 dataset, 2011.

[43] Phillip Isola, Joseph J Lim, and Edward H Adelson. Discovering states and transformations in image collections. In *Proceedings of the IEEE conference on computer vision and pattern recognition*, pages 1383–1391, 2015.

[44] Khoi Pham, Kushal Kafle, Zhe Lin, Zhihong Ding, Scott Cohen, Quan Tran, and Abhinav Shrivastava. Learning to predict visual attributes in the wild. In *Proceedings of the IEEE/CVF conference on computer vision and pattern recognition*, pages 13018–13028, 2021.

[45] Khoi Pham, Kushal Kafle, Zhe Lin, Zhihong Ding, Scott Cohen, Quan Tran, and Abhinav Shrivastava. Improving closed and open-vocabulary attribute prediction using transformers. In *European Conference on Computer Vision*, pages 201–219. Springer, 2022.

[46] Jacob Devlin, Ming-Wei Chang, Kenton Lee, and Kristina Toutanova. Bert: Pre-training of deep bidirectional transformers for language understanding. *arXiv preprint arXiv:1810.04805*, 2018.

[47] Ashish Vaswani, Noam Shazeer, Niki Parmar, Jakob Uszkoreit, Llion Jones, Aidan N Gomez, Łukasz Kaiser, and Illia Polosukhin. Attention is all you need. *Advances in neural information processing systems*, 30, 2017.

[48] Prannay Khosla, Piotr Teterwak, Chen Wang, Aaron Sarna, Yonglong Tian, Phillip Isola, Aaron Maschinot, Ce Liu, and Dilip Krishnan. Supervised contrastive learning. *Advances in neural information processing systems*, 33:18661–18673, 2020.

[49] Yanwei Fu, Timothy M Hospedales, Tao Xiang, and Shaogang Gong. Transductive multi-view zero-shot learning. *IEEE transactions on pattern analysis and machine intelligence*, 37(11):2332–2345, 2015.

[50] David F Gleich and Lek-heng Lim. Rank aggregation via nuclear norm minimization. In *Proceedings of the 17th ACM SIGKDD international conference on Knowledge discovery and data mining*, pages 60–68, 2011.

[51] Christoph H Lampert, Hannes Nickisch, and Stefan Harmeling. Attribute-based classification for zero-shot visual object categorization. *IEEE transactions on pattern analysis and machine intelligence*, 36(3):453–465, 2013.

[52] Maria-Elena Nilsback and Andrew Zisserman. Automated flower classification over a large number of classes. In *2008 Sixth Indian conference on computer vision, graphics & image processing*, pages 722–729. IEEE, 2008.

[53] Genevieve Patterson and James Hays. Sun attribute database: Discovering, annotating, and recognizing scene attributes. In *2012 IEEE conference on computer vision and pattern recognition*, pages 2751–2758. IEEE, 2012.

[54] Subhransu Maji, Esa Rahtu, Juho Kannala, Matthew Blaschko, and Andrea Vedaldi. Fine-grained visual classification of aircraft. *arXiv preprint arXiv:1306.5151*, 2013.

[55] Jonathan Krause, Michael Stark, Jia Deng, and Li Fei-Fei. 3d object representations for fine-grained categorization. In *Proceedings of the IEEE international conference on computer vision workshops*, pages 554–561, 2013.

[56] Patrick Helber, Benjamin Bischke, Andreas Dengel, and Damian Borth. Eurosat: A novel dataset and deep learning benchmark for land use and land cover classification. *IEEE Journal of Selected Topics in Applied Earth Observations and Remote Sensing*, 12(7):2217–2226, 2019.

[57] Mircea Cimpoi, Subhransu Maji, Iasonas Kokkinos, Sammy Mohamed, and Andrea Vedaldi. Describing textures in the wild. In *Proceedings of the IEEE conference on computer vision and pattern recognition*, pages 3606–3613, 2014.

[58] Khurram Soomro, Amir Roshan Zamir, and Mubarak Shah. Ucf101: A dataset of 101 human actions classes from videos in the wild. *arXiv preprint arXiv:1212.0402*, 2012.

[59] Lukas Bossard, Matthieu Guillaumin, and Luc Van Gool. Food-101–mining discriminative components with random forests. In *Computer Vision–ECCV 2014: 13th European Conference, Zurich, Switzerland, September 6-12, 2014, Proceedings, Part VI 13*, pages 446–461. Springer, 2014.

[60] Omkar M Parkhi, Andrea Vedaldi, Andrew Zisserman, and CV Jawahar. Cats and dogs. In *2012 IEEE conference on computer vision and pattern recognition*, pages 3498–3505. IEEE, 2012.

[61] Long Ouyang, Jeffrey Wu, Xu Jiang, Diogo Almeida, Carroll Wainwright, Pamela Mishkin, Chong Zhang, Sandhini Agarwal, Katarina Slama, Alex Ray, et al. Training language models to follow instructions with human feedback. *Advances in Neural Information Processing Systems*, 35:27730–27744, 2022.

[62] Yuxin Fang, Quan Sun, Xinggang Wang, Tiejun Huang, Xinlong Wang, and Yue Cao. Eva-02: A visual representation for neon genesis. *arXiv preprint arXiv:2303.11331*, 2023.

[63] Yonglong Tian, Dilip Krishnan, and Phillip Isola. Contrastive multiview coding. In *Computer Vision–ECCV 2020: 16th European Conference, Glasgow, UK, August 23–28, 2020, Proceedings, Part XI 16*, pages 776–794. Springer, 2020.

[64] An Yan, Yu Wang, Yiwu Zhong, Chengyu Dong, Zexue He, Yujie Lu, William Yang Wang, Jingbo Shang, and Julian McAuley. Learning concise and descriptive attributes for visual recognition. In *Proceedings of the IEEE/CVF International Conference on Computer Vision*, pages 3090–3100, 2023.

[65] Yue Yang, Artemis Panagopoulou, Shenghao Zhou, Daniel Jin, Chris Callison-Burch, and Mark Yatskar. Language in a bottle: Language model guided concept bottlenecks for interpretable image classification. In *Proceedings of the IEEE/CVF Conference on Computer Vision and Pattern Recognition*, pages 19187–19197, 2023.

[66] Victor Weixin Liang, Yuhui Zhang, Yongchan Kwon, Serena Yeung, and James Y Zou. Mind the gap: Understanding the modality gap in multi-modal contrastive representation learning. *Advances in Neural Information Processing Systems*, 35:17612–17625, 2022.

[67] Jason Wei and Kai Zou. Eda: Easy data augmentation techniques for boosting performance on text classification tasks. *arXiv preprint arXiv:1901.11196*, 2019.

[68] Yi Li, Hualiang Wang, Yiqun Duan, and Xiaomeng Li. Clip surgery for better explainability with enhancement in open-vocabulary tasks. *arXiv preprint arXiv:2304.05653*, 2023.

[69] Christoph Schuhmann, Romain Beaumont, Richard Vencu, Cade Gordon, Ross Wightman, Mehdi Cherti, Theo Coombes, Aarush Katta, Clayton Mullis, Mitchell Wortsman, et al. Laion-5b: An open large-scale dataset for training next generation image-text models. *Advances in Neural Information Processing Systems*, 35:25278–25294, 2022.

# A  Appendix

## A.1  Dataset Statistics

We present statistical information on the 12 datasets used in the experimental evaluations, as depicted in Table 5. The first four rows correspond to benchmark datasets widely employed in GZSL, with dataset splits provided in previous works[4]. The remaining eight datasets are commonly used for object recognition and encompass natural objects, textures, actions, scenes, etc. We adhere to a similar data split methodology in [4], dividing these eight datasets into seen and unseen classes.

Table 5: Detailed dataset statistics.

| Datasets | Classes | Seen classes | Unseen classes | Training & validation classes | Training & validation size | Testing size |
|---|---|---|---|---|---|---|
| AwA2 | 50 | 40 | 10 | 27+13 | 23,527 | 13,795 |
| CUB | 200 | 150 | 50 | 100+50 | 7,057 | 4,731 |
| FLO | 102 | 82 | 20 | 62+20 | 4,401 | 3,788 |
| SUN | 717 | 645 | 72 | 580+65 | 10,320 | 4,020 |
| FGVC-Aircraft | 100 | 80 | 20 | 55+25 | 6,400 | 3,600 |
| Country | 211 | 151 | 60 | 101+50 | 33,220 | 30,080 |
| StanfordCars | 196 | 136 | 60 | 95+41 | 7,891 | 8,294 |
| EuroSAT | 10 | 6 | 4 | 4+2 | 10,200 | 16,800 |
| DTD | 47 | 32 | 15 | 21+11 | 2,304 | 3,336 |
| UCF101 | 101 | 70 | 31 | 48+22 | 5,973 | 7,347 |
| Food101 | 101 | 72 | 29 | 49+23 | 50,400 | 50,600 |
| OxfordPets | 37 | 25 | 12 | 17+8 | 3,500 | 3,890 |

## A.2  Textual Description Exemplars

Exemplars of textual descriptions of 12 classes in the AwA2 dataset generated by ChatGPT are shown in Table 6. Notably, the generated descriptions almost cover the visual characteristics of the corresponding class.

## A.3  Additional Experiments

**Multiple Textual Description Evaluation.** Prior work [29] has demonstrated that using multiple textual descriptions for each class can capture its semantic characteristics more comprehensively, because these descriptions may complement one another. Therefore, we undertake an investigation into the influence of multiple descriptions on model performance on CUB. In this endeavor, we employ ChatGPT to generate five distinct descriptions for each class [29]. Subsequently, we randomly select between 1 and 5 descriptions, denoted as $n$, to form multiple descriptions for each class. To conduct experiments in the multiple text setting, we adapt the loss functions by integrating multi-view contrastive learning techniques [63]. As depicted in Fig. 5, we present the performance for each $n$ by repeating the sampling process 10 times. Notably, when $n$ is small, the recognition performance exhibits more significant fluctuations, attributed to substantial differences among each sampled text. As $n$ increases, the model's performance stability improves, and all three accuracy metrics gradually enhance, particularly the accuracy for recognizing unseen classes ($U$). The results underscore the advantages of leveraging multiple descriptions in improving overall model performance, with a notable impact on recognizing unseen classes.

**Textual Description Evaluation.** Although ChatGPT demonstrates significant generation capabilities, a comprehensive evaluation of the quality of the generated text descriptions has not yet been conducted. In this study, we approximately evaluate the quality of these text descriptions by calculating the cosine similarity between the description features and the corresponding attribute features (if available) for each class. As illustrated in Fig. 6, empirical analyses are performed on the AwA2 and CUB datasets, both of which provide ground-truth attribute annotations. The findings indicate a high similarity between the description features and the corresponding ground-truth attribute features, thereby suggesting a high quality of the generated descriptions.

**Correlation between Attributes.** We curate an extensive attribute vocabulary to comprehensively encapsulate various characteristics. As described in the main paper, TPR can still achieve relatively good results even if most of the items are removed from the attribute vocabulary. Here, we present

| Classes | Images | Textual descriptions |
|---|---|---|
| antelope |  | Antelopes are herbivorous mammals known for their slender bodies and long, curved horns. They are often found in grasslands and open savannas, where they graze on vegetation and use their speed and agility to escape predators. |
| grizzly bear |  | Grizzly bears are large, omnivorous mammals with a distinctive hump of muscle on their shoulders. They have shaggy fur that can range from blond to dark brown and are known for their strength and ability to catch fish from rivers. |
| killer whale |  | Killer whales, also known as orcas, are marine mammals known for their striking black and white coloration. They are highly intelligent and social, often traveling in pods. Orcas are skilled predators that feed on fish, seals, and even other whales. |
| beaver |  | Beavers are semi-aquatic rodents known for building dams and lodges. They have webbed feet and a flat, scaly tail, which they use for swimming and steering. Beavers are herbivores and primarily consume tree bark and aquatic plants. |
| dalmatian |  | Dalmatians are a breed of dog known for their distinctive black spots on a white coat. They are energetic and often associated with firehouses. Dalmatians are known for their friendly demeanor and strong guarding instincts. |
| persian cat |  | Persian cats are known for their long, luxurious fur and distinctive flat faces. They are a calm and gentle breed, often found as indoor pets. Persian cats require regular grooming due to their thick coats. |
| horse | 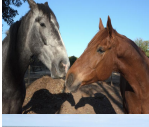 | Horses are large, hoofed mammals often used for riding, racing, and work. They come in various breeds and colors, with distinctive features such as a flowing mane and tail. Horses have played a crucial role in human history for transportation and agriculture. |
| german shepherd |  | German Shepherds are a breed of dog known for their intelligence, loyalty, and versatility. They have a distinctive appearance with a strong, muscular body and a double coat. German Shepherds are often used as working dogs in roles like police work and search and rescue. |
| blue whale |  | Blue whales are the largest animals on Earth, with a long and streamlined body that is predominantly blue-gray in color. They are filter feeders, using baleen plates to capture krill and other small marine organisms. |
| siamese cat |  | Siamese cats are known for their striking blue almond-shaped eyes and color-pointed fur, with a pale body and dark ears, face, paws, and tail. They are vocal and social cats with a strong bond to their owners. |
| skunk |  | Skunks are small mammals known for their distinctive black and white coloring and the ability to spray a foul-smelling liquid as a defense mechanism. They have bushy tails and are omnivorous, feeding on insects, small animals, and plants. |
| mole |  | Moles are burrowing mammals with velvety fur, small eyes, and powerful forelimbs equipped for digging. They primarily feed on insects and earthworms and create intricate tunnel systems underground. |

Table 6: Textual description demonstrations for 12 classes in AwA2, which are generated by ChatGPT. Best viewed in color.

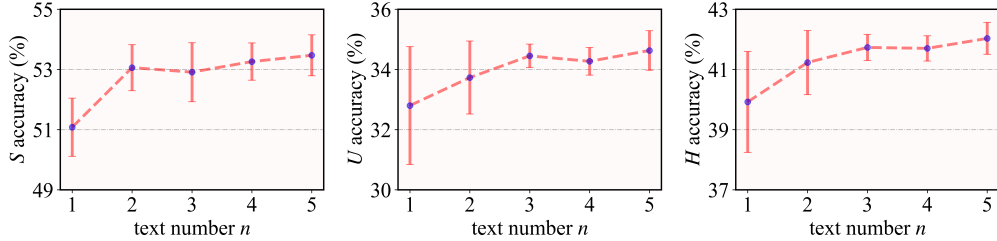

Figure 5: Effects of multiple textual descriptions on CUB. For each class, we randomly sample $n$ textual descriptions 10 times.

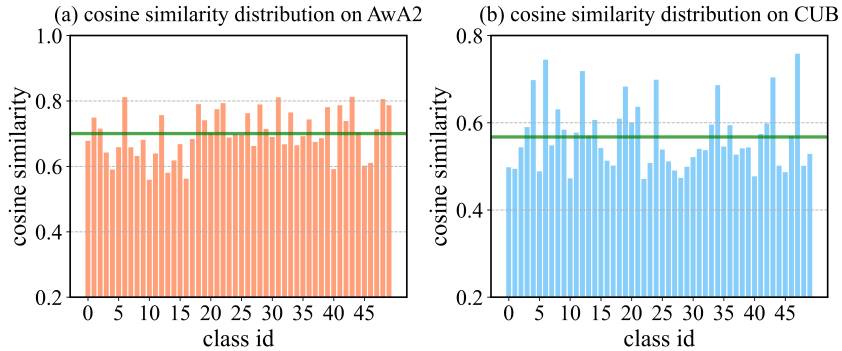

Figure 6: The cosine similarity between textual description features and corresponding attribute annotation features on AwA2 (left) and CUB (right).

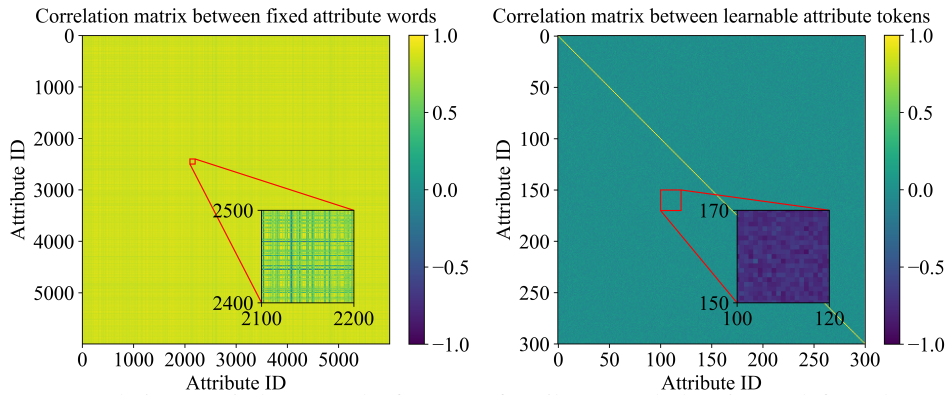

Figure 7: Correlation matrix between the features of attribute vocabulary items (left) and correlation matrix between the features of learnable attribute tokens. Best viewed in color.

the correlation matrix for the attribute vocabulary items and the correlation matrix for the learnable attribute tokens, as illustrated in Fig. 7. The figure demonstrates that many items in the attribute vocabulary are highly correlated, indicating redundancy within the vocabulary. Consequently, removing highly similar items does not significantly affect the final performance. In contrast, the correlation between learnable attribute tokens is much lower, requiring only a few hundred tokens to achieve performance gains. While it may seem possible to achieve the same recognition accuracy with a significantly smaller attribute vocabulary [64, 65], this paper primarily focuses on enhancing the fine-grained discrimination ability of features. We will address the issue of obtaining a more condensed attribute vocabulary in our subsequent work, potentially utilizing principal component analysis to achieve this objective.

**Visual Features Visualization.** In Fig. 8, we present the distribution of visual representations in the attribute space for the AwA2 and CUB datasets. For each dataset, the features of 500 samples are visualized with the t-SNE technique. The visualization reveals that samples from the same class are well clustered in the attribute space, while those from different classes are separated. In contrast,

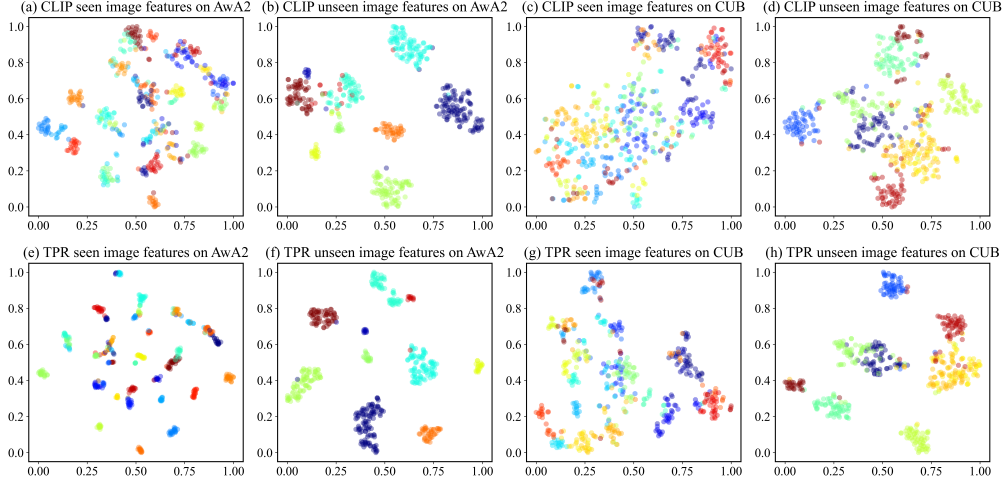

Figure 8: t-SNE visualization of visual representations on AwA2 and CUB. Instances of the same class are marked with the same color. Best viewed in color.

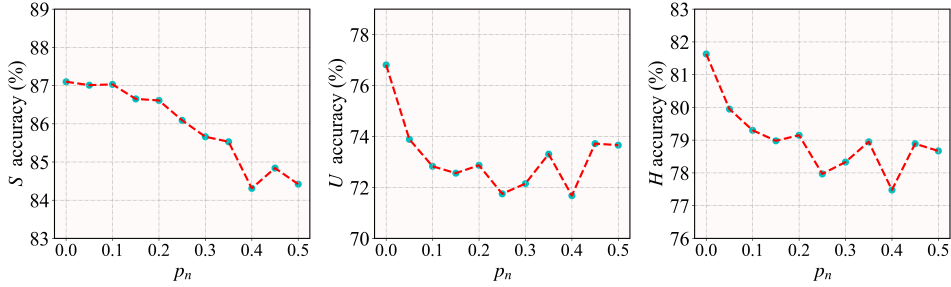

Figure 9: Impact of noisy textual descriptions on model performance on AwA2, and $p_n$ denotes the intensity of the imposed noise.

CLIP image features do not present good clustering characteristics. This observation aligns with findings in the literature [66], which indicate that CLIP image features tend to be more scattered and cluttered due to CLIP's objective of learning the association between images and their corresponding text, rather than developing compact image features. The visualization result demonstrates that TPR effectively aligns visual and textual features in a semantically structured attribute feature space.

**Robustness to Noisy Textual Descriptions.** To examine the impact of noisy text descriptions on TPR's performance, we introduce perturbations into the generated text descriptions. Specifically, we utilize the natural language perturbation tool proposed by EDA [67], which produces noisy text through four operations: **synonym replacement**, **random insertion**, **random swap**, and **random deletion**. Let $p_n$ denote the probability of applying these operations. These noisy descriptions are then used as the textual input for TPR, while the rest of the network remains consistent with the configuration described in the main paper. The experimental results, shown in Fig. 9, indicate that TPR maintains robustness against textual perturbations. As the noise intensity $p_n$ increases, the recognition performance of unseen classes significantly degrades, whereas the performance of seen classes gradually decreases. Notably, even under a 50% noise intensity, TPR consistently achieves a performance exceeding 78% on the $H$ metric.

**Visualization of Multimodal Features in the Attribute Space.** We present the distribution of visual features ($g_v$) and corresponding textual features ($g_l$) in the attribute space. For clarity, we display the feature values for 14 attributes across 10 categories in AwA2, as illustrated in Fig. 10. The visual feature values are averaged for each class. The distributions of visual and textual features for each class exhibit substantial similarity, thereby facilitating zero-shot recognition. Furthermore, these representation values align with the actual characteristics of each class, such as their visual appearance. For example, the *persian cat* class prominently features the attribute *with brown eyes*.

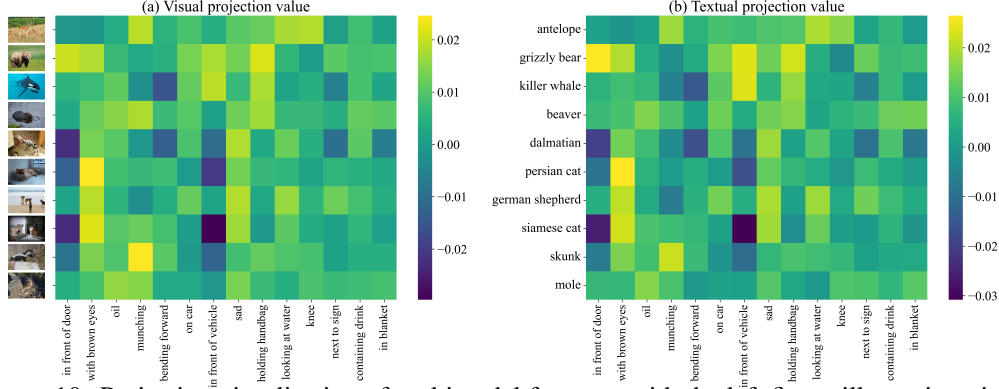

Figure 10: Projection visualization of multimodal features, with the left figure illustrating visual projection and the right figure illustrating textual projection. Best viewed in color.

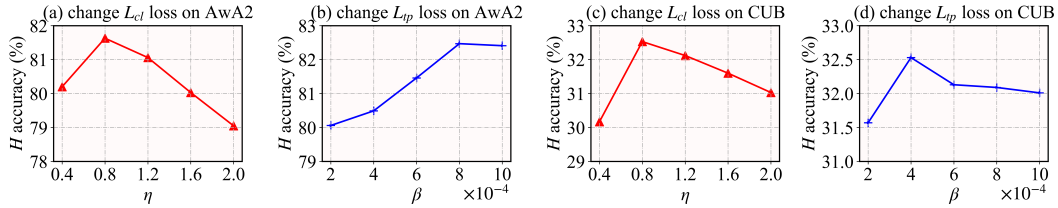

Figure 11: Impacts of the loss weights $\eta$ and $\beta$.

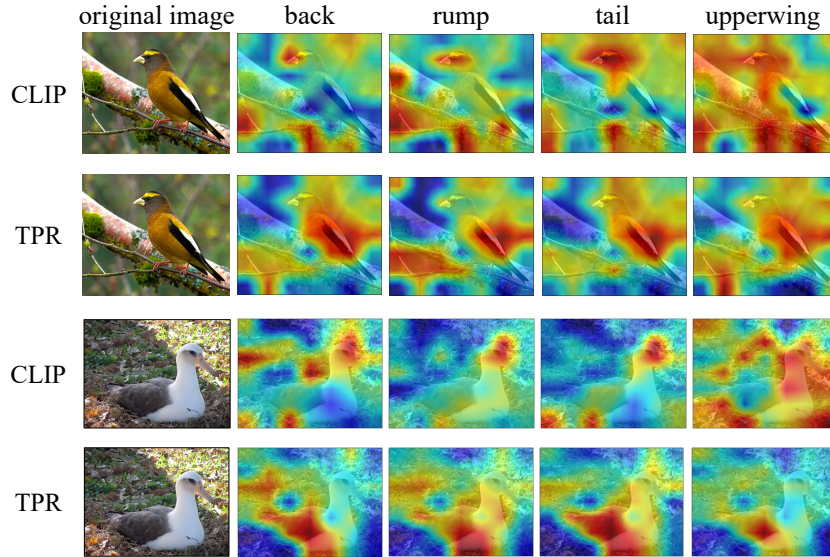

Figure 12: Textual respondence visualization. We present the response distribution to specific text within an image by using CLIP surgery [68]. The top row denotes the query text, with each subsequent row illustrating the heatmap distribution of responses from CLIP and our proposed method, respectively. Best viewed in color.

**Hyperparameter Sensitivity.** We investigate the impact of hyperparameter settings, specifically the loss weights $\eta$ and $\beta$, on model performance on the AwA2 and CUB datasets. As illustrated in Fig. 11, $H$ initially improves and subsequently declines as both $\eta$ and $\beta$ increase, with the maximum fluctuation being approximately 3%. TPR exhibits robustness to variations in $\eta$ and $\beta$ within a reasonable range.

**Textual Respondence Visualization.** In Fig. 12, we present a heatmap of textual descriptions in images using CLIP Surgery [68]. The color intensity represents the response value of the region to the respective text, with redder colors indicating higher response values. As demonstrated, TPR

Table 7: Ablation on backbones and pretraining data scale.

| Pretraining data | Backbone | AwA2 | | | CUB | | | FLO | | |
|---|---|---|---|---|---|---|---|---|---|---|
| | | $S$ | $U$ | $H$ | $S$ | $U$ | $H$ | $S$ | $U$ | $H$ |
| 400M | RN50 | 81.64 | 67.86 | 74.11 | 33.22 | 23.41 | 27.47 | 62.76 | 58.69 | 60.66 |
| | RN101 | 87.20 | 71.73 | 78.71 | 36.78 | 29.15 | 32.52 | 66.29 | 63.68 | 64.96 |
| | ViT-B/16 | **89.34** | **83.15** | **86.13** | **46.21** | **35.71** | **40.29** | 80.04 | 65.47 | **72.02** |
| | ViT-B/32 | 87.10 | 76.81 | 81.63 | 41.22 | 26.87 | 32.53 | 77.58 | 64.52 | 70.45 |
| 2B | ViT-B/16 | **90.77** | **81.86** | **86.08** | **58.56** | **48.44** | **53.02** | **82.02** | 65.51 | **72.84** |
| | ViT-B/32 | 86.80 | 78.41 | 82.39 | 47.05 | 37.02 | 41.44 | 81.62 | 64.78 | 72.24 |

Table 8: Results of TPR with text descriptions and TPR with ground-truth attributes. The experiments are conducted on both textual descriptions and attribute annotations to validate TPR's compatibility.

| Method | AwA2 | | | CUB | | | SUN | | |
|---|---|---|---|---|---|---|---|---|---|
| | $S$ | $U$ | $H$ | $S$ | $U$ | $H$ | $S$ | $U$ | $H$ |
| attribute | 76.73 | 63.50 | 69.49 | 24.39 | 22.74 | 23.54 | 42.98 | 41.60 | 42.28 |
| text | 87.10 | 76.81 | 81.63 | 41.22 | 26.87 | 32.53 | 50.47 | 45.40 | 47.80 |

localizes objects of interest, which helps to minimize background interference. In contrast, the heatmap generated by CLIP is more dispersed and tends to capture cluttered objects. However, TPR may exhibit insufficient focus; for instance, when the query is "back", regions outside the back still show large response values. We defer addressing this issue to future work, which will explore the potential use of attention mechanisms to refine focus on specific regions.

**Backbone Networks and Pretraining Data Scale.** In Table 7, we delve into the comparison of four distinct variants of the CLIP visual backbone, each characterized by a considerable number of parameters. Specifically, the ResNet50 model comprises 102M parameters, the ResNet101 model comprises 120M parameters, the ViT-B/16 model comprises 150M parameters, and the ViT-B/32 model consists of 151M parameters. It is noteworthy that as the number of model parameters increases, there is a substantial improvement in all three accuracy metrics across all three datasets. These results underscore the robust generalizability of TPR across different backbone networks. Subsequently, we examine the impact of the amount of pretraining data on model performance by extracting visual and linguistic features using Laion-CLIP (2B pretraining data) [69]. As depicted in Table 7, Laion-CLIP outperforms CLIP when employing both ViT-B/16 and ViT-B/32 backbones. These findings suggest that larger pretraining datasets generally result in improved model performance.

**Attribute-based GZSL.** In addition to utilizing pure text-based descriptions for GZSL, TPR can also leverage attribute annotations (if available). Specifically, we utilize the CLIP text encoder to extract attribute-annotated features for each category. Simultaneously, the image encoder is utilized to extract visual features for each image. The CLIP text and image encoders are kept frozen. The structure of TPR, except for the modified semantic inputs, remains consistent with the description provided in the main paper. The results presented in Table 8 underscore the compatibility of TPR with attribute-annotation-based GZSL.

